# Learning from Multiple Partially Observed Views – an Application to Multilingual Text Categorization

**Massih R. Amini**
Interactive Language Technologies Group
National Research Council Canada
Massih-Reza.Amini@cnrc-nrc.gc.ca

**Nicolas Usunier**
Laboratoire d'Informatique de Paris 6
Université Pierre et Marie Curie, France
Nicolas.Usunier@lip6.fr

**Cyril Goutte**
Interactive Language Technologies Group
National Research Council Canada
Cyril.Goutte@cnrc-nrc.gc.ca

## Abstract

We address the problem of learning classifiers when observations have multiple views, some of which may not be observed for all examples. We assume the existence of view generating functions which may complete the missing views in an approximate way. This situation corresponds for example to learning text classifiers from multilingual collections where documents are not available in all languages. In that case, Machine Translation (MT) systems may be used to translate each document in the missing languages. We derive a generalization error bound for classifiers learned on examples with multiple artificially created views. Our result uncovers a trade-off between the size of the training set, the number of views, and the quality of the view generating functions. As a consequence, we identify situations where it is more interesting to use multiple views for learning instead of classical single view learning. An extension of this framework is a natural way to leverage unlabeled multi-view data in semi-supervised learning. Experimental results on a subset of the Reuters RCV1/RCV2 collections support our findings by showing that additional views obtained from MT may significantly improve the classification performance in the cases identified by our trade-off.

## 1   Introduction

We study the learning ability of classifiers trained on examples generated from different sources, but where some observations are partially missing. This problem occurs for example in non-parallel multilingual document collections, where documents may be available in different languages, but each document in a given language may not be translated in all (or any) of the other languages.

Our framework assumes the existence of view generating functions which may approximate missing examples using the observed ones. In the case of multilingual corpora these view generating functions may be Machine Translation systems which for each document in one language produce its translations in all other languages. Compared to other multi-source learning techniques [6], we address a different problem here by transforming our initial problem of learning from partially observed examples obtained from multiple sources into the classical multi-view learning. The contributions of this paper are twofold. We first introduce a supervised learning framework in which we define different multi-view learning tasks. Our main result is a generalization error bound for classifiers trained over multi-view observations. From this result we induce a trade-off between the number of training examples, the number of views and the ability of view generating functions to

produce accurate additional views. This trade-off helps us identify situations in which artificially generated views may lead to substantial performance gains. We then show how the agreement of classifiers over their class predictions on unlabeled training data may lead to a much tighter trade-off. Experiments are carried out on a large part of the Reuters RCV1/RCV2 collections, freely available from Reuters, using 5 well-represented languages for text classification. Our results show that our approach yields improved classification performance in both the supervised and semi-supervised settings.

In the following two sections, we first define our framework, then the learning tasks we address. Section 4 describes our trade-off bound in the Empirical Risk Minimization (ERM) setting, and shows how and when the additional, artificially generated views may yield a better generalization performance in a supervised setting. Section 5 shows how to exploit these results when additional unlabeled training data are available, in order to obtain a more accurate trade-off. Finally, section 6 describes experimental results that support this approach.

## 2 Framework and Definitions

In this section, we introduce basic definitions and the learning objectives that we address in our setting of artificially generated representations.

### 2.1 Observed and Generated Views

A *multi-view observation* is a sequence $\mathbf{x} \stackrel{def}{=} (x^1, ..., x^V)$, where different *views* $x^v$ provide a representation of the same object in different sets $\mathcal{X}_v$. A typical example is given in [3] where each Web-page is represented either by its textual content (first view) or by the anchor texts which point to it (second view). In the setting of multilingual classification, each view is the textual representation of a document written in a given language (e.g. English, German, French).

We consider binary classification problems where, given a multi-view observation, some of the views are not observed (we obviously require that at least one view is observed). This happens, for instance, when documents may be available in different languages, yet a given document may only be available in a single language. Formally, our observations $\mathbf{x}$ belong to the input set $\mathcal{X} \stackrel{def}{=} (\mathcal{X}_1 \cup \{\bot\}) \times ... \times (\mathcal{X}_V \cup \{\bot\})$, where $x^v = \bot$ means that the $v$-th view is not observed. In binary classification, we assume that examples are pairs $(\mathbf{x}, y)$, with $y \in \mathcal{Y} \stackrel{def}{=} \{0, 1\}$, drawn according to a fixed, but unknown distribution $\mathcal{D}$ over $\mathcal{X} \times \mathcal{Y}$, such that $\mathbb{P}_{(\mathbf{x},y)\sim\mathcal{D}} (\forall v : x^v = \bot) = 0$ (at least one view is available). In multilingual text classification, a *parallel corpus* is a dataset where all views are always observed (i.e. $\mathbb{P}_{(\mathbf{x},y)\sim\mathcal{D}} (\exists v : x^v = \bot) = 0$), while a *comparable corpus* is a dataset where only one view is available for each example (i.e. $\mathbb{P}_{(\mathbf{x},y)\sim\mathcal{D}} (|\{v : x^v \neq \bot\}| \neq 1) = 0$).

For a given observation $\mathbf{x}$, the views $v$ such that $x^v \neq \bot$ will be called the *observed views*. The originality of our setting is that we assume that we have *view generating functions* $\Psi_{v \to v'} : \mathcal{X}_v \to \mathcal{X}_{v'}$ which take as input a given view $x^v$ and output an element of $\mathcal{X}_{v'}$, that we assume is *close* to what $x^{v'}$ would be if it was observed. In our multilingual text classification example, the view generating functions are Machine Translation systems. These generating functions can then be used to create surrogate observations, such that all views are available. For a given partially observed $\mathbf{x}$, the *completed* observation $\underline{\mathbf{x}}$ is obtained as:

$$\forall v, \underline{x}^v = \begin{cases} x^v & \text{if } x^v \neq \bot \\ \Psi_{v' \to v}(x^{v'}) & \text{otherwise, where } v' \text{ is such that } x^{v'} \neq \bot \end{cases} \quad (1)$$

In this paper, we focus on the case where only *one* view is observed for each example. This setting corresponds to the problem of learning from *comparable corpora*, which will be the focus of our experiments. Our study extends to the situation where two or more views may be observed in a straightforward manner. Our setting differs from previous multi-view learning studies [5] mainly on the straightforward generalization to more than two views and the use of view generating functions to induce the missing views from the observed ones.

## 2.2 Learning objective

The learning task we address is to find, in some predefined classifier set $\mathcal{C}$, the stochastic classifier $c$ that minimizes the classification error on multi-view examples (with, potentially, unobserved views) drawn according to some distribution $\mathcal{D}$ as described above. Following the standard multi-view framework, in which all views are observed [3, 13], we assume that we are given $V$ *deterministic* classifier sets $(\mathcal{H}_v)_{v=1}^V$, each working on one specific view[1]. That is, for each view $v$, $\mathcal{H}_v$ is a set of functions $h_v : \mathcal{X}_v \to \{0, 1\}$. The final set of classifiers $\mathcal{C}$ contains *stochastic* classifiers, whose output only depends on the outputs of the view-specific classifiers. That is, associated to a set of classifiers $\mathcal{C}$, there is a function $\Phi_\mathcal{C} : (\mathcal{H}_v)_{v=1}^V \times \mathcal{X} \to [0, 1]$ such that:

$$\mathcal{C} = \{\mathbf{x} \mapsto \Phi_\mathcal{C}(h_1, ..., h_V, \mathbf{x}) \, | \forall v, h_v \in \mathcal{H}_v \}$$

For simplicity, in the rest of the paper, when the context is clear, the function $\mathbf{x} \mapsto \Phi_\mathcal{C}(h_1, ..., h_V, \mathbf{x})$ will be denoted by $c_{h_1, ..., h_V}$. The overall objective of learning is therefore to find $c \in \mathcal{C}$ with low generalization error, defined as:

$$\epsilon(c) = \mathop{\mathbb{E}}_{(\mathbf{x}, y) \sim \mathcal{D}} e\left(c, (\mathbf{x}, y)\right) \tag{2}$$

where $e$ is a pointwise error, for instance the $0/1$ loss: $e(c, (\mathbf{x}, y)) = c(\mathbf{x})(1 - y) + (1 - c(\mathbf{x}))y$.

In the following sections, we address this learning task in our framework in terms of supervised and semi-supervised learning.

## 3 Supervised Learning Tasks

We first focus on the supervised learning case. We assume that we have a training set $\mathcal{S}$ of $m$ examples drawn i.i.d. according to a distribution $\mathcal{D}$, as presented in the previous section. Depending on how the generated views are used at both training and test stages, we consider the following learning scenarios:

- **Baseline:** This setting corresponds to the case where each view-specific classifier is trained using the corresponding observed view on the training set, and prediction for a test example is done using the view-specific classifier corresponding to the observed view:

$$\forall v, h_v \in \operatorname*{arg\,min}_{h \in \mathcal{H}_v} \sum_{(\mathbf{x}, y) \in S : x^v \neq \perp} e(h, (x^v, y)) \tag{3}$$

In this case we pose $\forall \mathbf{x}, c^b_{h_1, ..., h_V}(\mathbf{x}) = h_v(x^v)$, where $v$ is the observed view for $\mathbf{x}$. Notice that this is the most basic way of learning a text classifier from a comparable corpus.

- **Generated Views as Additional Training Data:** The most natural way to use the generated views for learning is to use them as additional training material for the view-specific classifiers:

$$\forall v, h_v \in \operatorname*{arg\,min}_{h \in \mathcal{H}_v} \sum_{(\mathbf{x}, y) \in S} e(h, (\underline{x}^v, y)) \tag{4}$$

with $\underline{\mathbf{x}}$ defined by eq. (1). Prediction is still done using the view-specific classifiers corresponding to the observed view, i.e. $\forall \mathbf{x}, c^b_{h_1, ..., h_V}(\mathbf{x}) = h_v(x^v)$. Although the test set distribution is a subdomain of the training set distribution [2], this mismatch is (hopefully) compensated by the addition of new examples.

- **Multi-view Gibbs Classifier:** In order to avoid the potential bias introduced by the use of generated views only during training, we consider them also during testing. This becomes a standard multi-view setting, where generated views are used exactly as if they were observed. The view-specific classifiers are trained exactly as above (eq. 4), but the prediction is carried out with respect to the probability distribution of classes, by estimating the probability of class membership in class 1 from the mean prediction of each view-specific classifier:

$$\forall \mathbf{x}, c^{mg}_{h_1, ..., h_V}(\mathbf{x}) = \frac{1}{V} \sum_{v=1}^V h_v(\underline{x}^v) \tag{5}$$

**- Multi-view Majority Voting:** With view generating functions involved in training and test, a natural way to obtain a (generally) deterministic classifier with improved performance is to take the majority vote associated with the Gibbs classifier. The view-specific classifiers are again trained as in eq. 4, but the final prediction is done using a majority vote:

$$\forall \mathbf{x}, c_{h_1,\ldots,h_V}^{mv}(\mathbf{x}) = \left\{ \begin{array}{ll} \frac{1}{2} & \text{if } \sum_{v=1}^{V} h_v(\underline{x}^v) = \frac{V}{2} \\ \mathrm{I}\left(\sum_{v=1}^{V} h_v(\underline{x}^v) > \frac{V}{2}\right) & \text{otherwise} \end{array} \right. \tag{6}$$

Where $I(.)$ is the indicator function. The classifier outputs either the majority voted class, or either one of the classes with probability $1/2$ in case of a tie.

## 4 The trade-offs with the ERM principle

We now analyze how the generated views can improve generalization performance. Essentially, the trade-off is that generated views offer additional training material, therefore potentially *helping* learning, but can also be of lower quality, which may *degrade* learning.

The following theorem sheds light on this trade-off by providing bounds on the baseline vs. multi-view strategies. Note that such trade-offs have already been studied in the literature, although in different settings (see e.g. [2, 4]). Our first result is the following theorem. The notion of function class capacity used here is the *empirical Rademacher complexity* [1]. Proof is given in the supplementary material.

**Theorem 1** *Let $\mathcal{D}$ be a distribution over $\mathcal{X} \times \mathcal{Y}$, satisfying $\mathbb{P}_{(\mathbf{x},y)\sim\mathcal{D}}\left(|\{v : x^v \neq \perp\}| \neq 1\right) = 0$. Let $\mathcal{S} = ((\mathbf{x}_i, y_i))_{i=1}^{m}$ be a dataset of $m$ examples drawn i.i.d. according to $\mathcal{D}$. Let $e$ be the $0/1$ loss, and let $(\mathcal{H}_v)_{v=1}^{V}$ be the view-specific deterministic classifier sets. For each view $v$, denote $e \circ \mathcal{H}_v \stackrel{def}{=} \{(x^v, y) \mapsto e(h, (x^v, y))|h \in \mathcal{H}_v\}$, and denote , for any sequence $\mathcal{S}^v \in (\mathcal{X}_v \times \mathcal{Y})^{m_v}$ of size $m_v$, $\hat{\mathcal{R}}_{m_v}(e \circ \mathcal{H}_v, \mathcal{S}^v)$ the empirical Rademacher complexity of $e \circ \mathcal{H}_v$ on $\mathcal{S}^v$. Then, we have:*

**Baseline setting:** *for all $1 > \delta > 0$, with probability at least $1 - \delta$ over $\mathcal{S}$:*

$$\epsilon(c_{h_1,\ldots,h_V}^{b}) \leq \inf_{h_v' \in \mathcal{H}_v} \left[\epsilon(c_{h_1',\ldots,h_V'}^{b})\right] + 2\sum_{v=1}^{V} \frac{m_v}{m}\hat{\mathcal{R}}_{m_v}(e \circ \mathcal{H}_v, \mathcal{S}^v) + 6\sqrt{\frac{\ln(2/\delta)}{2m}}$$

*where, for all $v$, $\mathcal{S}^v \stackrel{def}{=} \{(x_i^v, y_i)|i = 1..m \text{ and } x_i^v \neq \perp\}$, $m_v = |\mathcal{S}^v|$ and $h_v \in \mathcal{H}_v$ is the classifier minimizing the empirical risk on $\mathcal{S}^v$.*

**Multi-view Gibbs classification setting:** *for all $1 > \delta > 0$, with probability at least $1 - \delta$ over $\mathcal{S}$:*

$$\epsilon(c_{h_1,\ldots,h_V}^{mg}) \leq \inf_{h_v' \in \mathcal{H}_v} \left[\epsilon(c_{h_1',\ldots,h_V'}^{b})\right] + \frac{2}{V}\sum_{v=1}^{V} \hat{\mathcal{R}}_m(e \circ \mathcal{H}_v, \underline{\mathcal{S}}^v) + 6\sqrt{\frac{\ln(2/\delta)}{2m}} + \eta$$

*where, for all $v$, $\underline{\mathcal{S}}^v \stackrel{def}{=} \{(\underline{x}_i^v, y_i)|i = 1..m\}$, $h_v \in \mathcal{H}_v$ is the classifier minimizing the empirical risk on $\underline{\mathcal{S}}^v$, and*

$$\eta = \inf_{h_v' \in \mathcal{H}_v} \left[\epsilon(c_{h_1',\ldots,h_V'}^{mg})\right] - \inf_{h_v' \in \mathcal{H}_v} \left[\epsilon(c_{h_1',\ldots,h_V'}^{b})\right] \tag{7}$$

This theorem gives us a rule for whether it is preferable to learn only with the observed views (the baseline setting) or preferable to use the view-generating functions in the multi-view Gibbs classification setting: we should use the former when $2\sum_v \frac{m_v}{m}\hat{\mathcal{R}}_{m_v}(e \circ \mathcal{H}_v, \mathcal{S}^v) < \frac{2}{V}\sum_v \hat{\mathcal{R}}_m(e \circ \mathcal{H}_v, \underline{\mathcal{S}}^v) + \eta$, and the latter otherwise.

Let us first explain the role of $\eta$ (Eq. 7). The difference between the two settings is in the train and test distributions for the view-specific classifiers. $\eta$ compares the best achievable error for each of the distribution. $\inf_{h_v' \in \mathcal{H}_v} \left[\epsilon(c_{h_1',\ldots,h_V'}^{b})\right]$ is the best achievable error in the baseline setting (i.e. without generated views), with the automatically generated views, the best achievable error becomes $\inf_{h_v' \in \mathcal{H}_v} \left[\epsilon(c_{h_1',\ldots,h_V'}^{mg})\right]$.

Therefore $\eta$ measures the loss incurred by using the view generating functions. In a favorable situation, the quality of the generating functions will be sufficient to make $\eta$ small.

The terms depending on the complexity of the class of functions may be better explained using orders of magnitude. Typically, the Rademacher complexity for a sample of size $n$ is usually of order $O(\frac{1}{\sqrt{n}})$ [1].

Assuming, for simplicity, that all empirical Rademacher complexities in Theorem 1 are approximately equal to $d/\sqrt{n}$, where $n$ is the size of the sample on which they are computed, and assuming that $m_v = m/V$ for all $v$. The trade-off becomes:

*Choose the Multi-view Gibbs classification setting when:* $d\left(\sqrt{\frac{V}{m}} - \frac{1}{\sqrt{m}}\right) > \eta$

This means that we expect important performance gains when the number of examples is small, the generated views of sufficiently high quality for the given classification task, and/or there are many views available. Note that our theoretical framework does not take the quality of the MT system in a standard way: in our setup, a good translation system is (roughly) one which generates bag-of-words representations that allow to correctly discriminate between classes.

**Majority voting** One advantage of the multi-view setting at prediction time is that we can use a majority voting scheme, as described in Section 2. In such a case, we expect that $\epsilon(c^{mv}_{h'_1,...,h'_V}) \leq \epsilon(c^{mg}_{h'_1,...,h'_V})$ if the view-specific classifiers are not correlated in their errors. It can not be guaranteed in general, though, since, in general, we can not prove any better than $\epsilon(c^{mv}_{h'_1,...,h'_V}) \leq 2\epsilon(c^{mg}_{h'_1,...,h'_V})$ (see e.g. [9]).

## 5 Agreement-Based Semi-Supervised Learning

One advantage of the multi-view settings described in the previous section is that unlabeled training examples may naturally be taken into account in a semi–supervised learning scheme, using existing approaches for multi-view learning (e.g. [3]).

In this section, we describe how, under the framework of [11], the supervised learning trade-off presented above can be improved using extra unlabeled examples. This framework is based on the notion of *disagreement* between the various view-specific classifiers, defined as the expected variance of their outputs:

$$\mathbb{V}(h_1, ..., h_V) \overset{def}{=} \underset{(\mathbf{x},y)\sim\mathcal{D}}{\mathbb{E}}\left[\frac{1}{V}\sum_v h_v(\underline{x}^v)^2 - \left(\frac{1}{V}\sum_v h_v(\underline{x}^v)\right)^2\right] \tag{8}$$

The overall idea is that a set of good view-specific classifiers should agree on their predictions, making the expected variance small. This notion of disagreement has two key advantages. First, it does not depend on the true class labels, making its estimation easy over a large, unlabeled training set. The second advantage is that if, during training, it turns out that the view-specific classifiers have a disagreement of at most $\mu$ on the unlabeled set, the set of possible view-specific classifiers that needs be considered in the supervised learning stage is reduced to:

$$\mathcal{H}^*_v(\mu) \overset{def}{=} \{h'_v \in \mathcal{H}_v \,|\, \forall v' \neq v, \exists h'_{v'} \in \mathcal{H}_{v'}, \mathbb{V}(h'_1, ..., h'_V) \leq \mu\}$$

Thus, the more the various view-specific classifiers tend to agree, the smaller the possible set of functions will be. This suggests a simple way to do semi-supervised learning: the unlabeled data can be used to choose, among the classifiers minimizing the empirical risk on the labeled training set, those with best generalization performance (by choosing the classifiers with highest agreement on the unlabeled set). This is particularly interesting when the number of labeled examples is small, as the train error is usually close to $0$.

Theorem 3 of [11] provides a theoretical value $B(\epsilon, \delta)$ for the minimum number of unlabeled examples required to estimate Eq. 8 with precision $\epsilon$ and probability $1 - \delta$ (this bound depends on $\{\mathcal{H}_v\}_{v=1..V}$). The following result gives a tighter bound of the generalization error of the multi-view Gibbs classifier when unlabeled data are available. The proof is similar to Theorem 4 in [11].

**Proposition 2** *Let $0 \leq \mu \leq 1$ and $0 < \delta < 1$. Under the conditions and notations of Theorem 1, assume furthermore that we have access to $u \geq B(\mu/2, \delta/2)$ unlabeled examples drawn i.i.d. according to the marginal distribution of $\mathcal{D}$ on $\mathcal{X}$.*

*Then, with probability at least $1 - \delta$, if the empirical risk minimizers $h_v \in \arg\min_{h \in \mathcal{H}_v} \sum_{(\underline{x}^v, y) \in \mathcal{S}^v} e(h, (\underline{x}^v, y))$ have a disagreement less than $\mu/2$ on the unlabeled set, we have:*

$$\epsilon(c^{mg}_{h_1,\ldots,h_V}) \leq \inf_{h'_v \in \mathcal{H}_v} \left[ \epsilon(c^b_{h'_1,\ldots,h'_V}) \right] + \frac{2}{V} \sum_{v=1}^{V} \hat{\mathcal{R}}_m(e \circ \mathcal{H}^*_v(\mu), \underline{\mathcal{S}}^v) + 6\sqrt{\frac{\ln(4/\delta)}{2m}} + \eta$$

We can now rewrite the trade-off between the baseline setting and the multi-view Gibbs classifier, taking semi-supervised learning into account. Using orders of magnitude, and assuming that for each view, $\hat{\mathcal{R}}_m(e \circ \mathcal{H}^*_v(\mu), \underline{\mathcal{S}}^v)$ is $O(d_u/\sqrt{m})$, with the proportional factor $d_u \ll d$, the trade-off becomes:

*Choose the mutli-view Gibbs classification setting when: $d\sqrt{V/m} - d_u/\sqrt{m} > \eta$.*

Thus, the improvement is even more important than in the supervised setting. Also note that the more views we have, the greater the reduction in classifier set complexity should be.

Notice that this semi-supervised learning principle enforces agreement between the view specific classifiers. In the extreme case where they almost always give the same output, majority voting is then nearly equivalent to the Gibbs classifier (when all voters agree, any vote is equal to the majority vote). We therefore expect the majority vote and the Gibbs classifier to yield similar performance in the semi-supervised setting.

# 6    Experimental Results

In our experiments, we address the problem of learning document classifiers from a comparable corpus. We build the comparable corpus by sampling parts of the Reuters RCV1 and RCV2 collections [12, 14]. We used newswire articles written in 5 languages, `English`, `French`, `German`, `Italian` and `Spanish`. We focused on 6 relatively populous classes: `C15`, `CCAT`, `E21`, `ECAT`, `GCAT`, `M11`.

For each language and each class, we sampled up to 5000 documents from the RCV1 (for `English`) or RCV2 (for other languages). Documents belonging to more than one of our 6 classes were assigned the label of their smallest class. This resulted in 12-30K documents per language, and 11-34K documents per class (see Table 1). In addition, we reserved a test split containing 20% of the documents (respecting class and language proportions) for testing. For each document, we indexed the text appearing in the title (*headline* tag), and the body (*body* tags) of each article. As preprocessing, we lowercased, mapped digits to a single `digit` token, and removed non alphanumeric tokens. We also filtered out function words using a stop-list, as well as tokens occurring in less than 5 documents.

Documents were then represented as a bag of words, using a TFIDF-based weighting scheme. The final vocabulary size for each language is given in table 1. The artificial views were produced using

Table 1: Distribution of documents over languages and classes in the comparable corpus.

| Language | # docs | (%) | # tokens | Class | Size (all lang.) | (%) |
|---|---|---|---|---|---|---|
| English | $18,758$ | $16.78$ | $21,531$ | C15 | $18,816$ | $16.84$ |
| French | $26,648$ | $23.45$ | $24,893$ | CCAT | $21,426$ | $19.17$ |
| German | $29,953$ | $26.80$ | $34,279$ | E21 | $13,701$ | $12.26$ |
| Italian | $24,039$ | $21.51$ | $15,506$ | ECAT | $19,198$ | $17.18$ |
| Spanish | $12,342$ | $11.46$ | $11,547$ | GCAT | $19,178$ | $17.16$ |
| Total | $111,740$ | | | M11 | $19,421$ | $17.39$ |

PORTAGE, a statistical machine translation system developed at NRC [15]. Each document from the comparable corpus was thus translated to the other $4$ languages.[2]

For each class, we set up a binary classification task by using all documents from that class as positive examples, and all others as negative. We first present experimental results obtained in supervised learning, using various amounts of labeled examples. We rely on linear `SVM` models as base classifiers, using the `SVM-Perf` package [8]. For comparisons, we employed the four learning strategies described in section 3: $1-$ the single-view baseline $sv_b$ (Eq. 3), $2-$ generated views as additional training data $gv_b$ (Eq. 4), $3-$ multi-view Gibbs $mv_g$ (Eq. 5), and $4-$ multi-view majority voting $mv_m$ (Eq. 6). Recall that the second setting, $gv_b$, is the most straightforward way to train and test classifiers when additional examples are available (or generated) from different sources. It can thus be seen as a baseline approach, as opposed to the last two strategies ($mv_g$ and $mv_m$), where view-specific classifiers are both trained and tested over both original and translated documents. Note also that in our case ($V = 5$ views), additional training examples obtained from machine translation represent $4$ times as many labeled examples as the original texts used to train the baseline $sv_b$. All test results were averaged over $10$ randomly sampled training sets.

Table 2: Test classification accuracy and $F_1$ in the supervised setting, for both baselines ($sv_b$, $gv_b$), Gibbs ($mv_g$) and majority voting ($mv_w$) strategies, averaged over $10$ random sets of $10$ labeled examples per view. $\downarrow$ indicates statistically significantly worse performance that the best result, according to a Wilcoxon rank sum test ($p < 0.01$) [10].

| Strategy | C15 | | CCAT | | E21 | | ECAT | | GCAT | | M11 | |
|---|---|---|---|---|---|---|---|---|---|---|---|---|
| | Acc. | $F_1$ | Acc. | $F_1$ | Acc. | $F_1$ | Acc. | $F_1$ | Acc. | $F_1$ | Acc. | $F_1$ |
| $sv_b$ | $.559^\downarrow$ | $.388^\downarrow$ | $.639^\downarrow$ | $.403^\downarrow$ | $.557^\downarrow$ | $.294^\downarrow$ | $.579^\downarrow$ | $.374^\downarrow$ | $.800^\downarrow$ | $.501^\downarrow$ | $.651^\downarrow$ | $.483^\downarrow$ |
| $gv_b$ | $.705$ | $.474^\downarrow$ | $.691^\downarrow$ | $.464^\downarrow$ | $.665^\downarrow$ | $.351^\downarrow$ | $.623^\downarrow$ | $.424^\downarrow$ | $.835^\downarrow$ | $.595^\downarrow$ | $.786^\downarrow$ | $.589^\downarrow$ |
| $mv_g$ | $.693^\downarrow$ | $.494^\downarrow$ | $.681^\downarrow$ | $.445^\downarrow$ | $.665^\downarrow$ | $.375^\downarrow$ | $.620^\downarrow$ | $.420^\downarrow$ | $.834^\downarrow$ | $.594^\downarrow$ | $.787^\downarrow$ | $.600^\downarrow$ |
| $mv_m$ | **.716** | **.521** | **.708** | **.478** | **.693** | **.405** | **.636** | **.441** | **.860** | **.642** | **.820** | **.644** |

Results obtained in a supervised setting with only $10$ labeled documents per language for training are summarized in table 2. All learning strategies using the generated views during training outperform the single-view baseline. This shows that, although imperfect, artificial views do bring additional information that compensates the lack of labeled data. Although the multi-view Gibbs classifier predicts based on a translation rather than the original in $80\%$ of cases, it produces almost identical performance to the $gv_b$ run (which only predicts using the original text). These results indicate that the translation produced by our MT system is of sufficient quality for indexing and classification purposes. Multi-view majority voting reaches the best performance, yielding a $6 - 17\%$ improvement in accuracy over the baseline. A similar increase in performance is observed using $F_1$, which suggests that the multi-view `SVM` appropriately handles unbalanced classes.

Figure 1 shows the learning curves obtained on 3 classes, `C15`, `ECAT` and `M11`. These figures show that when there are enough labeled examples (around $500$ for these 3 classes), the artificial views do not provide any additional useful information over the original-language examples. These empirical results illustrate the trade-off discussed at the previous section. When there are sufficient original labeled examples, additional generated views do not provide more useful information for learning than what view-specific classifiers have available already.

We now investigate the use of unlabeled training examples for learning the view-specific classifiers. Our overall aim is to illustrate our findings from section 5. Recall that in the case where view-specific classifiers are in agreement over the class labels of a large number of unlabeled examples, the multi-view Gibbs and majority vote strategies should have the same performance. In order to enforce agreement between classifiers on the unlabeled set, we use a variant of the iterative co-training algorithm [3]. Given the view-specific classifiers trained on an initial set of labeled examples, we iteratively assign pseudo-labels to the unlabeled examples for which all classifier predictions agree. We then train new view-specific classifiers on the joint set of the original labeled examples, and those unanimously (pseudo-)labeled ones. Key differences between this algorithm and co-training are the number of views used for learning (5 instead of 2), and the use of unanimous and simultaneous labeling.

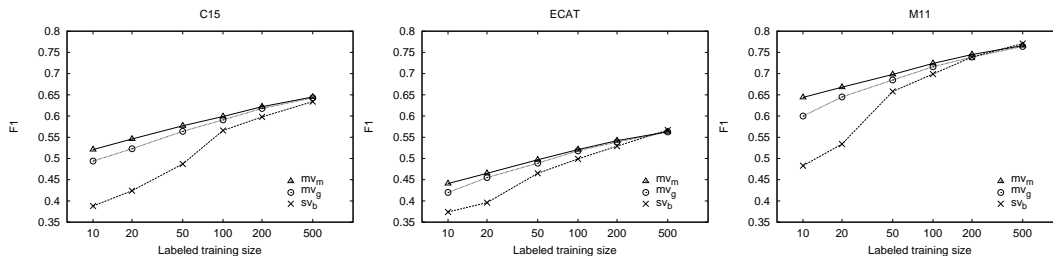

Figure 1: $F_1$ vs. size of the labeled training set for classes C15, ECAT and M11.

We call this iterative process *self-learning multiple-view algorithm*, as it also bears a similarity with the self-training paradigm [16]. Prediction from the multi-view SVM models obtained from this self-learning multiple-view algorithm is done either using Gibbs ($mv_g^s$) or majority voting ($mv_m^s$). These results are shown in table 3. For comparison we also trained a TSVM model [7] on each view separately, a semi-supervised equivalent to the single-view baseline strategy. Note that the TSVM model mostly out-performs the supervised baseline $sv_b$, although the $F_1$ suffers on some classes. This suggests that the TSVM has trouble handling unbalanced classes in this setting.

Table 3: Test classification accuracy and $F_1$ in the semi-supervised setting, for single-view TSVM and multi-view self-learning using either Gibbs ($mv_g^s$) or majority voting ($mv_m^s$), averaged over 10 random sets using 10 labeled examples per view to start. For comparison we provide the single-view baseline and multi-view majority voting performance for supervised learning.

| Strategy | C15 | | CCAT | | E21 | | ECAT | | GCAT | | M11 | |
|---|---|---|---|---|---|---|---|---|---|---|---|---|
| | Acc. | $F_1$ | Acc. | $F_1$ | Acc. | $F_1$ | Acc. | $F_1$ | Acc. | $F_1$ | Acc. | $F_1$ |
| $sv_b$ | $.559^{\downarrow}$ | $.388^{\downarrow}$ | $.639^{\downarrow}$ | $.403^{\downarrow}$ | $.557^{\downarrow}$ | $.294^{\downarrow}$ | $.579^{\downarrow}$ | $.374^{\downarrow}$ | $.800^{\downarrow}$ | $.501^{\downarrow}$ | $.651^{\downarrow}$ | $.483^{\downarrow}$ |
| $mv_m$ | $.716^{\downarrow}$ | $.521^{\downarrow}$ | $.708^{\downarrow}$ | $.478^{\downarrow}$ | $.693^{\downarrow}$ | $.405^{\downarrow}$ | $.636^{\downarrow}$ | $.441^{\downarrow}$ | $.860^{\downarrow}$ | $.642^{\downarrow}$ | $.820^{\downarrow}$ | $.644^{\downarrow}$ |
| TSVM | $.721^{\downarrow}$ | $.482^{\downarrow}$ | $.721^{\downarrow}$ | $.405^{\downarrow}$ | $.746^{\downarrow}$ | $.269^{\downarrow}$ | $.665^{\downarrow}$ | $.263^{\downarrow}$ | $.876^{\downarrow}$ | $.606^{\downarrow}$ | $.834^{\downarrow}$ | $.706^{\downarrow}$ |
| $mv_g^s$ | .772 | .586 | .762 | .538 | .765 | .470 | .691 | .504 | .903 | .729 | .900 | .764 |
| $mv_m^s$ | **.773** | **.589** | **.766** | **.545** | **.767** | **.473** | **.701** | **.508** | **.905** | **.734** | **.901** | **.766** |

The multi-view self-learning algorithm achieves the best classification performance in both accuracy and $F_1$, and significantly outperforms both the TSVM and the supervised multi-view strategy in all classes. As expected, the performance of both $mv_g^s$ and $mv_m^s$ strategies are similar.

## 7 Conclusion

The contributions of this paper are twofold. First, we proposed a bound on the risk of the Gibbs classifier trained over artificially completed multi-view observations, which directly corresponds to our target application of learning text classifiers from a comparable corpus. We showed that our bound may lead to a trade-off between the size of the training set, the number of views, and the quality of the view generating functions. Our result identifies in which case it is advantageous to learn with additional artificial views, as opposed to sticking with the baseline setting in which a classifier is trained over single view observations. This result leads to our second contribution, which is a natural way of using unlabeled data in semi-supervised multi-view learning. We showed that in the case where view-specific classifiers agree over the class labels of additional unlabeled training data, the previous trade-off becomes even much tighter. Empirical results on a comparable multilingual corpus support our findings by showing that additional views obtained using a Machine Translation system may significantly increase classification performance in the most interesting situation, when there are few labeled data available for training.

**Acknowlegdements** This work was supported in part by the IST Program of the European Community, under the PASCAL2 Network of Excellence, IST-2002-506778.

## Footnotes

[1]We assume deterministic view-specific classifiers for simplicity and with no loss of generality.

[2]The dataset is available from *http://multilingreuters.iit.nrc.ca/ReutersMultiLingualMultiView.htm*

# References

[1] P. L. Bartlett and S. Mendelson. Rademacher and gaussian complexities: risk bounds and structural results. *Journal of Machine Learning Research*, 3:463–482, 2003.

[2] J. Blitzer, K. Crammer, A. Kulesza, F. Pereira, and J. Wortman. Learning bounds for domain adaptation. In *NIPS*, 2007.

[3] A. Blum and T. M. Mitchell. Combining labeled and unlabeled sata with co-training. In *COLT*, pages 92–100, 1998.

[4] K. Crammer, M. Kearns, and J. Wortman. Learning from multiple sources. *Journal of Machine Learning Research*, 9:1757–1774, 2008.

[5] J. D. R. Farquhar, D. Hardoon, H. Meng, J. Shawe-Taylor, and S. Szedmak. Two view learning: Svm-2k, theory and practice. In *Advances in Neural Information Processing Systems 18*, pages 355–362. 2006.

[6] D. R. Hardoon, G. Leen, S. Kaski, and J. S.-T. (eds). Nips workshop on learning from multiple sources. 2008.

[7] T. Joachims. Transductive inference for text classification using support vector machines. In *ICML*, pages 200–209, 1999.

[8] T. Joachims. Training linear svms in linear time. In *Proceedings of the ACM Conference on Knowledge Discovery and Data Mining (KDD)*, pages 217–226, 2006.

[9] J. Langford and J. Shawe-taylor. Pac-bayes & margins. In *NIPS 15*, pages 439–446, 2002.

[10] E. Lehmann. *Nonparametric Statistical Methods Based on Ranks*. McGraw-Hill, New York, 1975.

[11] B. Leskes. The value of agreement, a new boosting algorithm. In *COLT*, pages 95–110, 2005.

[12] D. D. Lewis, Y. Yang, T. Rose, and F. Li. RCV1: A new benchmark collection for text categorization research. *Journal of Machine Learning Research*, 5:361–397, 2004.

[13] I. Muslea. *Active learning with multiple views*. PhD thesis, USC, 2002.

[14] Reuters. Corpus, volume 2, multilingual corpus, 1996-08-20 to 1997-08-19. 2005.

[15] N. Ueffing, M. Simard, S. Larkin, and J. H. Johnson. NRC's PORTAGE system for WMT. In *In ACL-2007 Second Workshop on SMT*, pages 185–188, 2007.

[16] X. Zhu. Semi-supervised learning literature survey. Technical report, Univ. Wisconsis, 2007.
